# A Bayesian Approach for Policy Learning from Trajectory Preference Queries

**Aaron Wilson** [*]
School of EECS
Oregon State University

**Alan Fern** [†]
School of EECS
Oregon State University

**Prasad Tadepalli** [‡]
School of EECS
Oregon State University

## Abstract

We consider the problem of learning control policies via trajectory preference queries to an expert. In particular, the agent presents an expert with short runs of a pair of policies originating from the same state and the expert indicates which trajectory is preferred. The agent's goal is to elicit a latent target policy from the expert with as few queries as possible. To tackle this problem we propose a novel Bayesian model of the querying process and introduce two methods that exploit this model to actively select expert queries. Experimental results on four benchmark problems indicate that our model can effectively learn policies from trajectory preference queries and that active query selection can be substantially more efficient than random selection.

## 1  Introduction

Directly specifying desired behaviors for automated agents is a difficult and time consuming process. Successful implementation requires expert knowledge of the target system and a means of communicating control knowledge to the agent. One way the expert can communicate the desired behavior is to directly demonstrate it and have the agent learn from the demonstrations, e.g. via imitation learning [15, 3, 13] or inverse reinforcement learning [12]. However, in some cases, like the control of complex robots or simulation agents, it is difficult to generate demonstrations of the desired behaviors. In these cases an expert may still recognize when an agent's behavior matches a desired behavior, or is close to it, even if it is difficult to directly demonstrate it. In such cases an expert may also be able to evaluate the relative qualities to the desired behavior of a pair of example trajectories and express a preference for one or the other.

Given this motivation, we study the problem of learning expert policies via trajectory preference queries to an expert. A *trajectory preference query (TPQ)* is a pair of short state trajectories originating from a common state. Given a TPQ the expert is asked to indicate which trajectory is most similar to the target behavior. The goal of our learner is to infer the target trajectory using as few TPQs as possible. Our first contribution (Section 3) is to introduce a Bayesian model of the querying process along with an inference approach for sampling policies from the posterior given a set of TPQs and their expert responses. Our second contribution (Section 4) is to describe two active query strategies that attempt to leverage the model in order to minimize the number of queries required. Finally, our third contribution (Section 5) is to empirically demonstrate the effectiveness of the model and querying strategies on four benchmark problems.

We are not the first to examine preference learning for sequential decision making. In the work of Cheng et al. [5] action preferences were introduced into the classification based policy iteration

---

[*]`wilsonaa@eecs.oregonstate.edu`
[†]`afern@eecs.oregonstate.edu`
[‡]`tadepall@eecs.oregonstate.edu`

framework. In this framework preferences explicitly rank state-action pairs according to their relative payoffs. There is no explicit interaction between the agent and domain expert. Further the approach also relies on knowledge of the reward function, while our work derives all information about the target policy by actively querying an expert. In more closely related to our work, Akraur et al. [1] consider the problem of learning a policy from expert queries. Similar to our proposal this work suggests presenting trajectory data to an informed expert. However, their queries require the expert to express preferences over approximate state visitation densities and to possess knowledge of the expected performance of demonstrated policies. Necessarily the trajectories must be long enough to adequately approximate the visitation density. We remove this requirement and only require short demonstrations; our expert assesses trajectory snippets not whole solutions. We believe this is valuable because pairs of short demonstrations are an intuitive and manageable object for experts to assess.

## 2   Preliminaries

We explore policy learning from expert preferences in the framework of Markov Decision Processes (MDP). An MDP is a tuple $(S, A, T, P_0, R)$ with state space $S$, action space $A$, state transition distribution $T$, which gives the probability $T(s, a, s')$ of transitioning to state $s'$ given that action $a$ is taken in state $s$. The initial state distribution $P_0(s_0)$ gives a probability distribution over initial states $s_0$. Finally the reward function $R(s)$ gives the reward for being in state $s$. Note that in this work, the agent will not be able to observe rewards and rather must gather all information about the quality of policies via interaction with an expert. We consider agents that select actions using a policy $\pi_\theta$ parameterized by $\theta$, which is a stochastic mapping from states to actions $P_\pi(a|s, \theta)$. For example, in our experiments, we use a log-linear policy representation, where the parameters correspond to coefficients of features defined over state-action pairs.

Agents acting in an MDP experience the world as a sequence of state-action pairs called a trajectory. We denote a $K$-length trajectory as $\xi = (s_0, a_0, ..., a_{K-1}, s_K)$ beginning in state $s_0$ and terminating after K steps. It follows from the definitions above that the probability of generating a K-length trajectory given that the agent executes policy $\pi_\theta$ starting from state $s_0$ is, $P(\xi|\theta, s_0) = \prod_{t=1}^{K} T(s_{t-1}, a_{t-1}, s_t) P_\pi(a_{t-1}|s_{t-1}, \theta)$. Trajectories are an important part of our query process. They are an intuitive means of communicating policy information. Trajectories have the advantage that the expert need not share a language with the agent. Instead the expert is only required to recognize differences in physical performances presented by the agent. For purposes of generating trajectories we assume that our learner is provided with a strong simulator (or generative model) of the MDP dynamics, which takes as input a start state $s$, a policy $\pi$, and a value $K$, and outputs a sampled length $K$ trajectory of $\pi$ starting in $s$.

In this work, we evaluate policies in an episodic setting where an episode starts by drawing an initial state from $P_0$ and then executing the policy for a finite horizon $T$. A policy's value is the expected total reward of an episode. The goal of the learner is to select a policy whose value is close to that of an expert's policy. Note, that our work is not limited to finite-horizon problems, but can also be applied to infinite-horizon formulations.

In order to learn a policy, the agent presents *trajectory preference queries (TPQs)* to the expert and receives responses back. A TPQ is a pair of length $K$ trajectories $(\xi_i, \xi_j)$ that originate from a common state $s$. Typically $K$ will be much smaller than the horizon $T$, which is important from the perspective of expert usability. Having been provided with a TPQ the expert gives a response $y$ indicating, which trajectory is preferred. Thus, each TPQ results in a training data tuple $(\xi_i, \xi_j, y)$. Intuitively, the preferred trajectory is the one that is most similar to what the expert's policy would have produced from the same starting state. As detailed more in the next section, this is modeled by assuming that the expert has a (noisy) evaluation function $f(.)$ on trajectories and the response is then given by $y = I(f(\xi_i) > f(\xi_j))$ (a binary indicator). We assume that the expert's evaluation function is a function of the observed trajectories and a latent target policy $\theta^*$.

## 3   Bayesian Model and Inference

In this section we first describe a Bayesian model of the expert response process, which will be used to: 1) Infer policies based on expert responses to TPQs, and 2) Guide the action selection of

TPQs. Next, we describe a posterior sampling method for this model which is used for both policy inference and TPQ selection.

## 3.1 Expert Response Model

The model for the expert response $y$ given a TPQ $(\xi_i, \xi_j)$ decomposes as follows

$$P(y|(\xi_i, \xi_j), \theta^*)P(\theta^*)$$

where $P(\theta^*)$ is a prior over the latent expert policy, and $P(y|(\xi_i, \xi_j), \theta^*)$ is a response distribution conditioned on the TPQ and expert policy. In our experiments we use a ridge prior in the form of a Gaussian over $\theta^*$ with diagonal covariance, which penalizes policies with large parameter values.

**Response Distribution.** The conditional response distribution is represented in terms of an *expert evaluation function* $f^*(\xi_i, \xi_j, \theta^*)$, described in detail below, which translates a TPQ and a candidate expert policy $\theta^*$ into a measure of preference for trajectory $\xi_i$ over $\xi_j$. Intuitively, $f^*$ measures the degree to which the policy $\theta^*$ agrees with $\xi_i$ relative to $\xi_j$. To translate the evaluation into an expert response we borrow from previous work [6]. In particular, we assume the expert response is given by the indicator $I(f^*(\xi_i, \xi_j, \theta^*) > \epsilon)$ where $\epsilon \sim N(0, \sigma_r^2)$. The indicator simply returns 1 if the condition is true, indicating $\xi_i$ is preferred, and zero otherwise. It follows that the conditional response distribution is given by:

$$P(y = 1|(\xi_i, \xi_j), \theta^*) = \int_{-\infty}^{+\infty} I(f^*(\xi_i, \xi_j, \theta^*) > \epsilon)N(\epsilon|0, \sigma_r^2)d\epsilon$$

$$= \Phi\left(\frac{f^*(\xi_i, \xi_j, \theta^*)}{\sigma_r}\right).$$

where $\Phi(.)$ denotes the cumulative distribution function of the normal distribution. This formulation allows the expert to err when demonstrated trajectories are difficult to distinguish as measured by the magnitude of the evaluation function $f^*$. We now describe the evaluation function in more detail.

**Evaluation Function.** Intuitively the evaluation function must combine distances between the query trajectories and trajectories generated by the latent target policy. We say that a latent policy and query trajectory are in agreement when they produce similar trajectories. The dissimilarity between two trajectories $\xi_i$ and $\xi_j$ is measured by the trajectory dissimilarity function

$$f(\xi_i, \xi_j) = \sum_{t=0}^{K} k([s_{i,t}, a_{i,t}], [s_{j,t}, a_{j,t}])$$

where the variables $[s_{i,t}, a_{i,t}]$ represent the values of the state-action pair at time step t in trajectory $i$ (similarly for $[s_{j,t}, a_{j,t}]$) and the function $k$ computes distances between state-action pairs. In our experiments, states and actions are represented by real-valued vectors and we use a simple function of the form: $k([s, a], [s', a']) = \|s - s'\| + \|a - a'\|$ though other more sophisticated comparison functions could be easily used in the model.

Given the trajectory comparison function, we now encode a dissimilarity measure between the latent target policy and an observed trajectory $\xi_i$. To do this let $\xi^*$ be a random variable ranging over length $k$ trajectories generated by target policy $\theta^*$ starting in the start state of $\xi_i$. The dissimilarity measure is given by:

$$d(\xi_i, \theta^*) = E[f(\xi_i, \xi^*)]$$

This function computes the expected dissimilarity between a query trajectory $\xi_i$ and the K-length trajectories generated by the latent policy from the same initial state.

Finally, the comparison function value $f^*(\xi_i, \xi_j, \theta^*) = d(\xi_j, \theta^*) - d(\xi_i, \theta^*)$ is the difference in computed values between the ith and jth trajectory. Larger values of $f^*$ indicate stronger preferences for trajectory $\xi_i$.

## 3.2 Posterior Inference

Given the definition of the response model, the prior distribution, and an observed data set $D = \{(\xi_i, \xi_j, y)\}$ of TPQs and responses the posterior distribution is,

$$P(\theta^*|D) \propto P(\theta^*) \prod_{(\xi_i, \xi_j, y) \in D} \Phi(z)^y (1 - \Phi(z))^{1-y},$$

where $z = \frac{d(\xi_j, \theta^*) - d(\xi_i, \theta^*)}{\sigma_r}$. This posterior distribution does not have a simple closed form and we must approximate it.

We approximate the posterior distribution using a set of posterior samples which we generate using a stochastic simulation algorithm called Hybrid Monte Carlo (HMC) [8, 2]. The HMC algorithm is an example of a Markov Chain Monte Carlo (MCMC) algorithm. MCMC algorithms output a sequence of samples from the target distribution. HMC has an advantage in our setting because it introduces auxiliary momentum variables proportional to the gradient of the posterior which guides the sampling process toward the modes of the posterior distribution.

To apply the HMC algorithm we must derive the gradient of the energy function $\bigtriangledown_{\theta^*} \log(P(D|\theta)P(\theta))$ as follows.

$$\frac{\partial}{\partial\theta_i^*} \log[P(\theta^*|D)] = \frac{\partial}{\partial\theta_i^*} \log[P(\theta^*)] + \sum_{(\xi_i,\xi_j,y)\in D} \frac{\partial}{\partial\theta_i^*} \log\left[\Phi\left(z\right)^y \left(1-\Phi\left(z\right)\right)^{1-y}\right]$$

The energy function decomposes into prior and likelihood components. Using our assumption of a Gaussian prior with diagonal covariance on $\theta^*$ the partial derivative of the prior component at $\theta_i^*$ is

$$\frac{\partial}{\partial\theta_i^*} \log[P(\theta^*)] = -\frac{(\theta_i^* - \mu)}{\sigma^2}.$$

Next, consider the gradient of the data log likelihood,

$$\sum_{(\xi_i,\xi_j,y)\in D} \frac{\partial}{\partial\theta_i^*} \log\left[\Phi(z)^y(1-\Phi(z))^{1-y}\right],$$

which decomposes into $|D|$ components each of which has a value dependent on $y$.

In what follows we will assume that $y = 1$ (It is straight forward to derive the second case). Recall that the function $\Phi(.)$ is the cumulative distribution function of $N(z; 0, \sigma_r^2)$, Therefore, the gradient of $\log(\Phi(z))$ is,

$$\frac{\partial}{\partial\theta_i^*} \log[\Phi(z)] = \frac{1}{\Phi(z)} \left(\frac{\partial}{\partial\theta_i^*}\Phi(z)\right) = \frac{1}{\Phi(z)} \left(\frac{\partial}{\partial\theta_i^*}z\right) N(z; 0, \sigma_r^2)$$

$$= \frac{1}{\Phi(z)} \left(\frac{1}{\sigma_r} \left(\frac{\partial}{\partial\theta_i^*}d(\xi_j, \theta^*) - \frac{\partial}{\partial\theta_i^*}d(\xi_i, \theta^*)\right) N(z; 0, \sigma_r^2)\right).$$

Rrecall the definition of $d(\xi, \theta^*)$ from above. After moving the derivative inside the integral the gradient of this function is

$$\frac{\partial}{\partial\theta_i^*} d(\xi, \theta^*) = -\int f(\xi, \xi^*)\frac{\partial}{\partial\theta_i^*}P(\xi^*|\theta^*)d\xi^* = -\int f(\xi, \xi^*)P(\xi^*|\theta^*)\frac{\partial}{\partial\theta_i^*} \log(P(\xi^*|\theta^*))d\xi^*$$

$$= -\int f(\xi, \xi^*)P(\xi^*|\theta^*) \sum_{k=1}^{K} \frac{\partial}{\partial\theta_i^*} \log(P_\pi(a_k|s_k, \theta^*))d\xi^*.$$

The final step follows from the definition of the trajectory density which decomposes under the log transformation. For purposes of approximating the gradient this integral must be estimated. We do this by generating $N$ sample trajectories from $P(\xi^*|\theta^*)$ and then compute the Monte-Carlo estimate $-\frac{1}{N}\sum_{l=1}^{N} f(\xi, \xi_l^*) \sum_{k=1}^{K} \frac{\partial}{\partial\theta_i^*} \log(P_\pi(a_k|s_k, \theta^*))$. We leave the definition of $\log(P_\pi(a_k|s_k, \theta^*))$ for the experimental results section where we describe a specific kind of stochastic policy space.

Given this gradient calculation, we can apply HMC in order to sample policy parameter vectors from the posterior distribution. This can be used for policy selection in a number of ways. For example, a policy could be formed via Bayesian averaging. In our experiments, we select a policy by generating a large set of samples and then select the sample maximizing the energy function.

## 4 Active Query Selection

Given the ability to perform posterior inference, the question now is how to collect a data set of TPQs and their responses. Unlike many learning problems, there is no natural distribution over TPQs to draw from, and thus, active selection of TPQs is essential. In particular, we want the learner to select TPQs for which the responses will be most useful toward the goal of learning the target policy. This selection problem is difficult due to the high dimensional continuous space of TPQs, where each TPQ is defined by an initial state and two trajectories originating from the state. To help overcome this complexity our algorithm assumes the availability of a distribution $\hat{P}_0$ over

candidate start states of TPQs. This distribution is intended to generate start states that are feasible and potentially relevant to a target policy. The distribution may incorporate domain knowledge to rule out unimportant parts of the space (e.g. avoiding states where the bicycle has crashed) or simply specify bounds on each dimension of the state space and generate states uniformly within the bounds. Given this distribution, we consider two approaches to actively generating TPQs for the expert.

## 4.1 Query by Disagreement

Our first approach *Query by Disagreement (QBD)* is similar to the well-known query-by-committee approach to active learning of classifiers [17, 9]. The main idea behind the basic query-by-committee approach is to generate a sequence of unlabeled examples from a given distribution and for each example sample a pair of classifiers from the current posterior. If the sampled classifiers disagree on the class of the example, then the algorithm queries the expert for the class label. This simple approach is often effective and has theoretical guarantees on its efficiency.

We can apply this general idea to select TPQs in a straightforward way. In particular, we generate a sequence of potential initial TPQ states from $\hat{P}_0$ and for each draw two policies $\theta_i$ and $\theta_j$ from the current posterior distribution $P(\theta^*|D)$. If the policies "disagree" on the state, then a query is posed based on trajectories generated by the policies. Disagreement on an initial state $s_0$ is measured according to the expected difference between $K$ length trajectories generated by $\theta_i$ and $\theta_j$ starting at $s_0$. In particular, the disagreement measure is $g = \int_{(\xi_i, \xi_j)} P(\xi_i|\theta_i, s_0, K) P(\xi_j|\theta_j, s_0, K) f(\xi_i, \xi_j)$, which we estimate via sampling a set of $K$ length trajectories from each policy. If this measure exceeds a threshold then TPQ is generated and given to the expert by running each policy for $K$ steps from the initial state. Otherwise a new initial state is generated. If no query is posed after a specified number of initial states, then the state and policy pair that generated the most disagreement are used to generate the TPQ. We set the threshold $t$ so that $\Phi(t/\sigma_r) = .95$.

This query strategy has the benefit of generating TPQs such that $\xi_i$ and $\xi_j$ are significantly different. This is important from a usability perspective, since making preference judgements between similar trajectories can be difficult for an expert and error prone. In practice we observe that the QBD strategy often generates TPQs based on policy pairs that are from different modes of the distribution, which is an intuitively appealing property.

## 4.2 Expected Belief Change

Another class of active learning approaches for classifiers is more selective than traditional query-by-committee. In particular, they either generate or are given an unlabeled dataset and then use a heuristic to select the most promising example to query from the entire set. Such approaches often outperform less selective approaches such as the traditional query-by-committee. In this same way, our second active learning approach for TPQs attempts to be more selective than the above QBD approach by generating a set of candidate TPQs and heuristically selecting the best among those candidates.

A set of candidate TPQs is generated by first drawing an initial state from from $\hat{P}_0$, sampling a pair of policies from the posterior, and then running the policies for K steps from the initial state. It remains to define the heuristic used to select the TPQ for presentation to the expert.

A truly Bayesian heuristic selection strategy should account for the overall change in belief about the latent target policy after adding a new data point. To represent the difference in posterior beliefs we use the variational distance between posterior based on the current data $D$ and the posterior based on the updated data $D \cup \{(\xi_i, \xi_j, y)\}$.

$$V(P(\theta|D) \parallel P(\theta|D \cup \{(\xi_i, \xi_j, y)\})) = \int |P(\theta|D) - P(\theta|D \cup \{(\xi_i, \xi_j, y)\})| d\theta.$$

By integrating over the entire latent policy space it accounts for the total impact of the query on the agent's beliefs.

The value of the variational distance depends on the response to the TPQ, which is unobserved at query selection time. Therefore, the agent computes the expected variational distance,

$$H(d) = \sum_{y \in 0,1} P(y|\xi_i, \xi_j, D) V(P(\theta|D) \parallel P(\theta|D \cup \{(\xi_i, \xi_j, y)\})).$$

Where $P(y|\xi_i, \xi_j, D) = \int P(y|\xi_i, \xi_j, \theta^*)P(\theta^*|D)d\theta^*$ is the predictive distribution and is straight-forwardly estimated using a set of posterior samples.

Finally, we specify a simple method of estimating the variational distance given a particular response. For this, we re-express the variational distance as an expectation with respect to $P(\theta|D)$,

$$V(P(\theta|D) \parallel P(\theta|D \cup d)) = \int |P(\theta|D) - P(\theta|D \cup d)| \, d\theta = \int \left| P(\theta|D) - P(\theta|D \cup d)\frac{P(\theta|D)}{P(\theta|D)} \right| d\theta$$

$$= \int P(\theta|D) \left| 1 - \frac{P(\theta|D \cup d)}{P(\theta|D)} \right| d\theta = \int P(\theta|D) \left| 1 - P(d|\theta)\frac{z_1}{z_2} \right| d\theta$$

where $z_1$ and $z_2$ are the normalizing constants of the posterior distributions. The final expression is a likelihood weighted estimate of the variational distance. We can estimate this value using Monte-Carlo over a set $S$ of policies sampled from the posterior,

$$V(P(\theta|D) \parallel P(\theta|D \cup (\xi_i, \xi_j, y))) \approx \sum_{\theta \in S} \left| 1 - \frac{z_1}{z_2} P(d|\theta) \right|$$

This leaves the computation of the ratio of normalizing constants $\frac{z_1}{z_2}$ which we estimate using Monte-Carlo based on a sample set of policies from the prior distribution, hence avoiding further posterior sampling.

Our basic strategy of using an information theoretic selection heuristic is similar to early work using Kullback Leibler Divergence ([7]) to measure the quality of experiments [11, 4]. Our approach differs in that we use a symmetric measure which directly computes differences in probability instead of expected differences in code lengths. The key disadvantage of this form of look-ahead query strategy (shared by other strategies of this kind) is the computational cost.

## 5  Empirical Results

Below we outline our experimental setup and present our empirical results on four standard RL benchmark domains.

### 5.1  Setup

If the posterior distribution focuses mass on the expert policy parameters the expected value of the MAP parameters will converge to the expected value of the expert's policy. Therefore, to examine the speed of convergence to the desired expert policy we report the performance of the MAP policy in the MDP task. We choose the MAP policy, maximizing $P(D|\theta)P(\theta)$, from the sample generated by our HMC routine. The expected return of the selected policy is estimated and reported. Note that no reward information is given to the learner and is used for evaluation only.

We produce an automated expert capable of responding to the queries produced by our agent. The expert knows a target policy, and compares, as described above, the query trajectories generated by the agent to the trajectories generated by the target policy. The expert stochastically produces a response based on its evaluations. Target policies are hand designed and produce near optimal performance in each domain.

In all experiments the agent executes a simple parametric policy, $P(a|s, \theta) = \frac{exp(\phi(s) \cdot \theta_a)}{\sum_{b \in A} exp(\phi(s) \cdot \theta_b)}$. The function $\phi(s)$ is a set of features derived from the current state s. The complete parameter vector $\theta$ is decomposed into components $\theta_a$ associated with each action $a$. The policy is executed by sampling an action from this distribution. The gradient of this action selection policy can be derived straightforwardly and substituted into the gradient of the energy function required by our HMC procedure.

We use the following values for the unspecified model parameters: $\sigma_r^2 = 1$, $\sigma^2 = 2$, $\mu = 0$. The value of $K$ used for TPQ trajectories was set to 10 for each domain except for Bicycle, for which we used $K = 300$. The Bicycle simulator uses a fine time scale, so that even $K = 300$ only corresponds to a few seconds of bike riding, which is quite reasonable for a TPQ.

For purposes of comparison we implement a simple random TPQ selection strategy (Denoted Random in the graphs below). The random strategy draws an initial TPQ state from $\hat{P}_0$ and then generates a trajectory pair by executing two policies drawn i.i.d. from the prior distribution $P(\theta)$. Thus, this approach does not use information about past query responses when selecting TPQs.

**Domains.** We consider the following benchmark domains.

*Acrobot.* The acrobot task simulates a two link under-actuated robot. One joint end, the "hands" of the robot, rotates around a fixed point. The mid joint associated with the "hips" attach the upper and lower links of the robot. To change the joint angle between the upper and lower links the agent applies torque at the hip joint. The lower link swings freely. Our expert knows a policy for swinging the acrobot into a balanced handstand. The acrobot system is defined by four continuous state variables $(\theta_1, \theta_2, \dot{\theta}_1, \dot{\theta}_2)$ representing the arrangement of the acrobot's joints and the changing velocities of the joint angles. The acrobot is controlled by a 12 dimensional softmax policy selecting between positive, negative, and zero torque to be applied at the hip joint. The feature vector $\phi(s)$ returns the vector of state variables. The acrobot receives a penalty on each step proportional to the distance between the foot and the target position for the foot.

*Mountain Car.* The mountain car domain simulates an underpowered vehicle which the agent must drive to the top of a steep hill. The state of the mountain car system is described by the location of the car $x$, and its velocity $v$. The goal of the agent controlling the mountain car system is to utilize the hills surrounding the car to generate sufficient energy to escape a basin. Our expert knows a policy for performing this escape. The agent's softmax control policy space has 16 dimensions and selects between positive and negative accelerations of the car. The feature vector $\phi(s)$ returns a polynomial expansion $(x, v, x^2, x^3, xv, x^2v, x^3v, v^2)$ of the state. The agent receives a penalty for every step taken to reach the goal.

*Cart Pole.* In the cart-pole domain the agent attempts to balance a pole fixed to a movable cart while maintaining the carts location in space. Episodes terminate if the pole falls or the cart leaves its specified boundary. The state space is composed of the cart velocity $v$, change in cart velocity $v'$, angle of the pole $\omega$, and angular velocity of the pole $\omega'$. The control policy has 12 dimensions and selects the magnitude of the change in velocity (positive or negative) applied to the base of the cart. The feature vector returns the state of the cart-pole. The agent is penalized for pole positions deviating from upright and for movement away from the midpoint.

*Bicycle Balancing.* Agents in the bicycle balancing task must keep the bicycle balanced for 30000 steps. For our experiments we use the simulator originally introduced in [14]. The state of the bicycle is defined by four variables $(\omega, \dot{\omega}, \nu, \dot{\nu})$. The variable $\omega$ is the angle of the bicycle with respect to vertical, and $\dot{\omega}$ is its angular velocity. The variable $\nu$ is the angle of the handlebars with respect to neutral, and $\dot{\nu}$ is the angular velocity. The goal of the agent is to keep the bicycle from falling. Falling occurs when $|\omega| > \pi/15$. We borrow the same implementation used in[10] including the discrete action set, the 20 dimensional feature space, and 100 dimensional policy. The agent selects from a discrete set of five actions. Each discrete action has two components. The first component is the torque applied to the handlebars $T \in (-1, 0, 1)$, and the second component is the displacement of the rider in the saddle $p \in (-.02, 0, .02)$. From these components five action tuples are composed $a \in ((-1, 0), (1, 0), (0, -.02), (0, .02), (0, 0))$. The agent is penalized proportional to the magnitude of $\omega$ at each step and receives a fixed penalty for falling.

We report the results of our experiments in Figure 1. Each graph gives the results for the TPQ selection strategies Random, Query-by-Disagreement (QBD), and Expected Belief Change (EBC). The average reward versus number of queries is provided for each selection strategy, where curves are averaged over 20 runs of learning.

## 5.2 Experiment Results

In all domains the learning algorithm successfully learns the target policy. This is true independent of the query selection procedure used. As can be seen our algorithm can successfully learn even from queries posed by Random. This demonstrates the effectiveness of our HMC inference approach.

Importantly, in some cases, the active query selection heuristics significantly improve the rate of convergence compared to Random. The value of the query selection procedures is particularly high in the Mountain Car and Cart Pole domains. In the Mountain Car domain more than 500 Random queries were needed to match the performance of 50 EBC queries. In both of these domains examining the generated query trajectories shows that the Random strategy tended to produce difficult to distinguish trajectory data and later queries tended to resemble earlier queries. This is due to "plateaus" in the policy space which produce nearly identical behaviors. Intuitively, the information content of queries selected by Random decreases rapidly leading to slower convergence. By

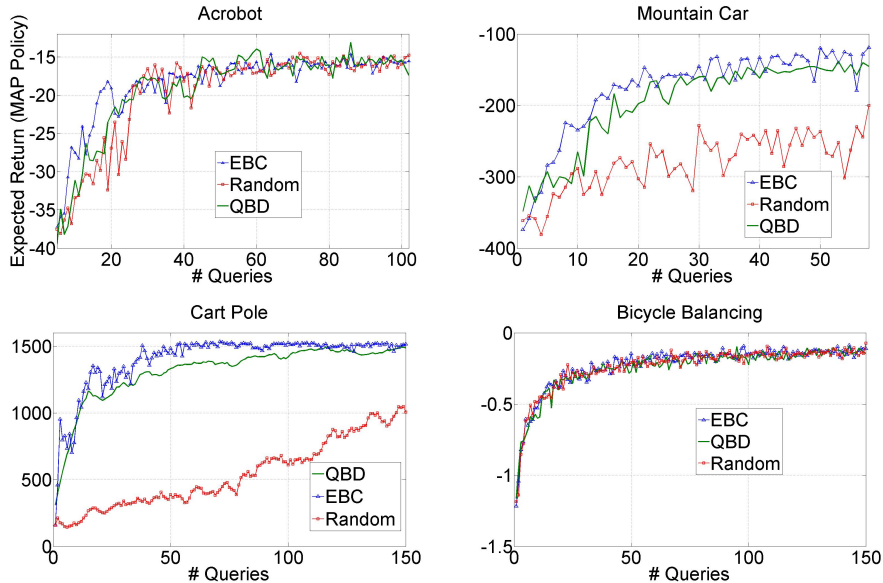

Figure 1: Results: We report the expected return of the MAP policy, sampled during Hybrid MCMC simulation of the posterior, as a function of the number of expert queries. Results are averaged over 50 runs. Query trajectory lengths: Acrobot $K = 10$, Mountain-Car $K = 10$, Cart-Pole $K = 20$, Bicycle Balancing $K = 300$.

comparison the selection heuristics ensure that selected queries have high impact on the posterior distribution and exhibit high query diversity.

The benefits of the active selection procedure diminish in the Acrobot and Bicycle domains. In both of these domains active selection performs only slightly better than Random. This is not the first time active selection procedures have shown performance similar to passive methods [16]. In Acrobot all of the query selection procedure quickly converge to the target policy (only 25 queries are needed for Random to identify the target). Little improvement is possible over this result. Similarly, in the bicycle domain the performance results are difficult to distinguish. We believe this is due to the length of the query trajectories (300) and the importance of the initial state distribution. Most bicycle configurations lead to out of control spirals from which no policy can return the bicycle to balanced. In these configurations inputs from the agent result in small impact on the observed state trajectory making policies difficult to distinguish. To avoid these cases in Bicycle the start state distribution $\hat{P}_0$ only generated initial states close to a balanced configuration. In these configurations poor balancing policies are easily distinguished from better policies and the better policies are not rare. These factors lead Random to be quite effective in this domain.

Finally, comparing the active learning strategies, we see that EBC has a slight advantage over QBD in all domains other than Bicycle. This agrees with prior active learning work, where more selective strategies tend to be superior in practice. The price that EBC pays for the improved performance is in computation time, as it is about an order of magnitude slower.

## 6 Summary

We examined the problem of learning a target policy via trajectory preference queries. We formulated a Bayesian model for the problem and a sampling algorithm for sampling from the posterior over policies. Two query selection methods were introduced, which heuristically select queries with an aim to efficiently identify the target. Experiments in four RL benchmarks indicate that our model and inference approach is able to infer quality policies and that the query selection methods are generally more effective than random selection.

### Acknowledgments

We gratefully acknowledge the support of ONR under grant number N00014-11-1-0106.

# References

[1] R. Akrour, M. Schoenauer, and M. Sebag. Preference-based policy learning. In Dimitrios Gunopulos, Thomas Hofmann, Donato Malerba, and Michalis Vazirgiannis, editors, *Proc. ECML/PKDD'11, Part I*, volume 6911 of *Lecture Notes in Computer Science*, pages 12–27. Springer, 2011.

[2] Christophe Andrieu, Nando de Freitas, Arnaud Doucet, and Michael I. Jordan. An introduction to mcmc for machine learning. *Machine Learning*, 50(1-2):5–43, 2003.

[3] Brenna D. Argall, Sonia Chernova, Manuela Veloso, and Brett Browning. A survey of robot learning from demonstration. *Robot. Auton. Syst.*, 57(5):469–483, May 2009.

[4] J M Bernardo. Expected information as expected utility. *Annals of Statistics*, 7(3):686–690, 1979.

[5] Weiwei Cheng, Johannes Fürnkranz, Eyke Hüllermeier, and Sang-Hyeun Park. Preference-based policy iteration: Leveraging preference learning for reinforcement learning. In *Proceedings of the 22nd European Conference on Machine Learning (ECML 2011)*, pages 312–327. Springer, 2011.

[6] Wei Chu and Zoubin Ghahramani. Preference learning with gaussian processes. In *Proceedings of the 22nd international conference on Machine learning*, ICML '05, pages 137–144, New York, NY, USA, 2005. ACM.

[7] Thomas M. Cover and Joy A. Thomas. *Elements of information theory*. Wiley-Interscience, New York, NY, USA, 1991.

[8] Simon Duane, A. D. Kennedy, Brian J. Pendleton, and Duncan Roweth. Hybrid monte carlo. *Physics Letters B*, 195(2):216 – 222, 1987.

[9] Yoav Freund, H. Sebastian Seung, Eli Shamir, and Naftali Tishby. Selective sampling using the query by committee algorithm. *Machine Learning*, 28(2-3):133–168, 1997.

[10] Michail G. Lagoudakis, Ronald Parr, and L. Bartlett. Least-squares policy iteration. *Journal of Machine Learning Research*, 4, 2003.

[11] D. V. Lindley. On a Measure of the Information Provided by an Experiment. *The Annals of Mathematical Statistics*, 27(4):986–1005, 1956.

[12] Andrew Y. Ng and Stuart J. Russell. Algorithms for inverse reinforcement learning. In *ICML*, pages 663–670, 2000.

[13] Bob Price and Craig Boutilier. Accelerating reinforcement learning through implicit imitation. *J. Artif. Intell. Res. (JAIR)*, 19:569–629, 2003.

[14] Jette Randløv and Preben Alstrøm. Learning to drive a bicycle using reinforcement learning and shaping. In *ICML*, pages 463–471, 1998.

[15] Stefan Schaal. Learning from demonstration. In *NIPS*, pages 1040–1046, 1996.

[16] Andrew I. Schein and Lyle H. Ungar. Active learning for logistic regression: an evaluation. *Mach. Learn.*, 68(3):235–265, October 2007.

[17] H. S. Seung, M. Opper, and H. Sompolinsky. Query by committee. In *Proceedings of the fifth annual workshop on Computational learning theory*, COLT '92, pages 287–294, New York, NY, USA, 1992. ACM.

